# Using Voice Transformations to Create Additional Training Talkers for Word Spotting

**Eric I. Chang and Richard P. Lippmann**
MIT Lincoln Laboratory
Lexington, MA 02173-0073, USA
eichang@sst.ll.mit.edu and rpl@sst.ll.mit.edu

## Abstract

Speech recognizers provide good performance for most users but the error rate often increases dramatically for a small percentage of talkers who are "different" from those talkers used for training. One expensive solution to this problem is to gather more training data in an attempt to sample these outlier users. A second solution, explored in this paper, is to artificially enlarge the number of training talkers by transforming the speech of existing training talkers. This approach is similar to enlarging the training set for OCR digit recognition by warping the training digit images, but is more difficult because continuous speech has a much larger number of dimensions (e.g. linguistic, phonetic, style, temporal, spectral) that differ across talkers. We explored the use of simple linear spectral warping to enlarge a 48-talker training data base used for word spotting. The average detection rate overall was increased by 2.9 percentage points (from 68.3% to 71.2%) for male speakers and 2.5 percentage points (from 64.8% to 67.3%) for female speakers. This increase is small but similar to that obtained by doubling the amount of training data.

## 1 INTRODUCTION

Speech recognizers, optical character recognizers, and other types of pattern classifiers used for human interface applications often provide good performance for most users. Performance is often, however, low and unacceptable for a small percentage of "outlier" users who are presumably not represented in the training data. One expensive solution to this problem is to obtain more training data in the hope of including users from these outlier

classes. Other approaches already used for speech recognition are to use input features and distance metrics that are relatively invariant to linguistically unimportant differences between talkers and to adapt a recognizer for individual talkers. Talker adaptation is difficult for word spotting and with poor outlier users because the recognition error rate is high and talkers often can not be prompted to recite standard phrases that can be used for adaptation. An alternative approach, that has not been fully explored for speech recognition, is to artificially expand the number of training talkers using voice transformations.

Transforming the speech of one talker to make it sound like that of another is difficult because speech varies across many difficult-to-measure dimensions including linguistic, phonetic, duration, spectra, style, and accent. The transformation task is thus more difficult than in optical character recognition where a small set of warping functions can be successfully applied to character images to enlarge the number of training images (Drucker, 1993). This paper demonstrates how a transformation accomplished by warping the spectra of training talkers to create more training data can improve the performance of a whole-word word spotter on a large spontaneous-speech data base.

## 2   BASELINE WORD SPOTTER

A hybrid radial basis function (RBF) – hidden Markov model (HMM) keyword spotter has been developed over the past few years that provides state-of-the-art performance for a whole-word word spotter on the large spontaneous-speech credit-card speech corpus. This system spots 20 target keywords, includes one general filler class, and uses a Viterbi decoding backtrace as described in (Chang, 1994) to backpropagate errors over a sequence of input speech frames. This neural network word spotter is trained on target and background classes, normalizes target outputs using the background output, and thresholds the resulting score to generate putative hits, as shown in Figure 1. Putative hits in this figure are input patterns which generate normalized scores above a threshold. The performance of this, and other spotting systems, is analyzed by plotting a detection versus false alarm rate curve. This curve is generated by adjusting the classifier output threshold to allow few or many putative hits. The figure of merit (FOM) is defined as the average keyword detection rate when the false alarm rate ranges from 1 to 10 false alarms per keyword per hour. The previous best FOM for this word spotter is 67.8% when trained using 24 male talkers and tested on 11 male talkers, and 65.9% when trained using 24 female talkers and tested on 11 female talkers. The overall FOM for all talkers is 66.3%.

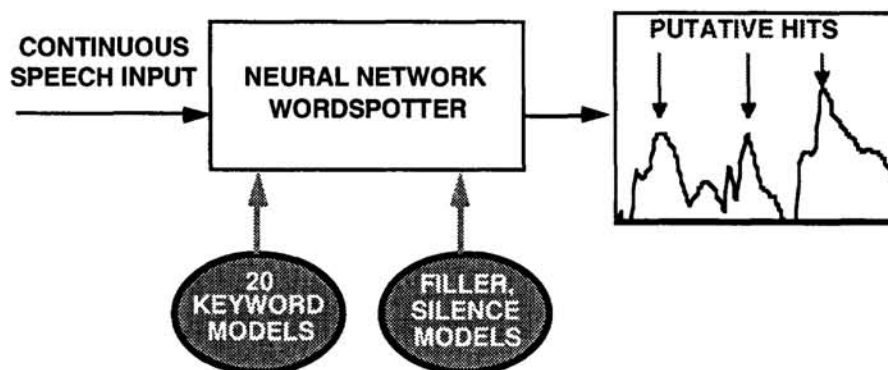

Figure 1:      Block diagram of neural network word spotter.

## 3  TALKER VARIABILITY

FOM scores of test talkers vary over a wide range. When training on 48 talkers and then performing testing on 22 talkers from the 70 conversations in the NIST Switchboard credit card database, the FOM of the test talkers varies from 16.7% to 100%. Most talkers perform well above 50%, but there are two female talkers with FOM's of 16.7% and 21.4%. The low FOM for individual speakers indicates a lack of training data with voice qualities that are similar to these test speakers.

## 4  CREATING MORE TRAINING DATA USING VOICE TRANSFORMATIONS

Talker adaptation is difficult for word spotting because error rates are high and talkers often can not be prompted to verify adaptation phrases. Our approach to increasing performance across talkers uses voice transformation techniques to generate more varied training examples of keywords as shown in Figure 2. Other researchers have used talker transformation techniques to produce more natural synthesized speech (Iwahashi, 1994, Mizuno, 1994), but using talker transformation techniques to generate more training data is novel.

We have implemented a new voice transformation technique which utilizes the Sinusoidal Transform Analysis/Synthesis System (STS) described in (Quatieri, 1992). This technique attempts to transform one talker's speech pattern to that of a different talker. The STS generates a 512 point spectral envelope of the input speech 100 times a second and also separates pitch and voicing information. Separation of vocal tract characteristic and pitch information has allowed the implementation of pitch and time transformations in previous work (Quatieri, 1992). The system has been modified to generate and accept a spectral en-

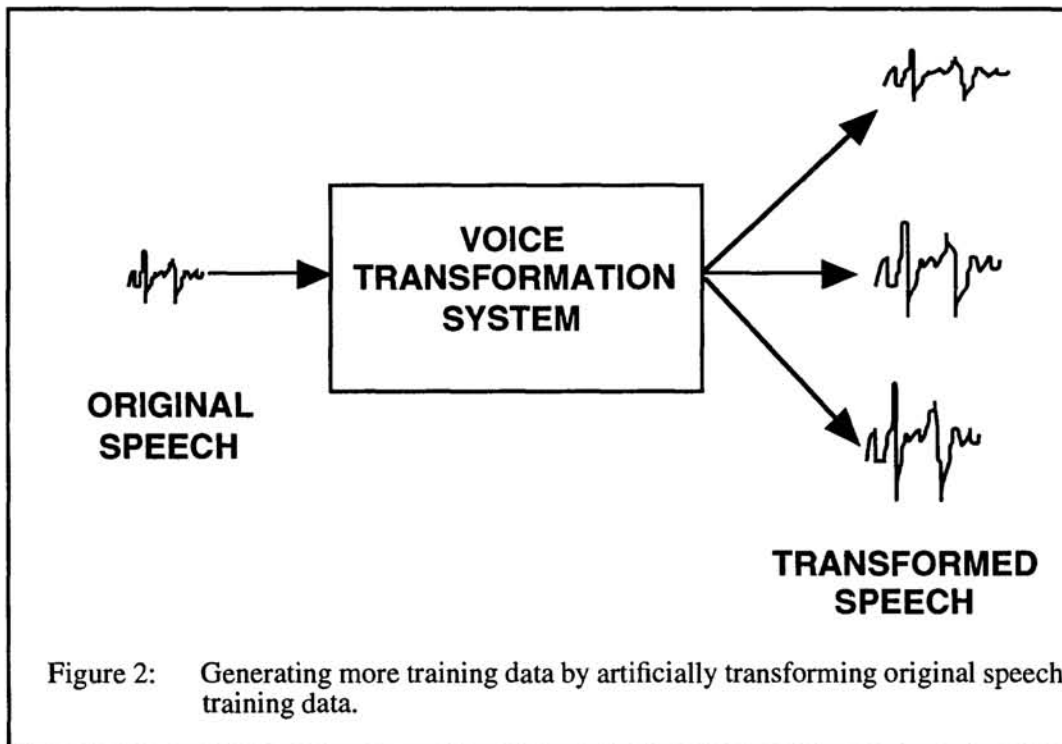

Figure 2:     Generating more training data by artificially transforming original speech training data.

velope file from an input speech sample. We informally explored different techniques to transform the spectral envelope to generate more varied training examples by listening to transformed speech. This resulted in the following algorithm that transforms a talker's voice by scaling the spectral envelope of training talkers.

1.   Training conversations are upsampled from 8000 Hz to 10,000 Hz to be compatible with existing STS coding software.

2.   The STS system processes the upsampled files and generates a 512 point spectral envelope of the input speech waveform at a frame rate of 100 frames a second and with a window length of approximately 2.5 times the length of each pitch period.

3.   A new spectral envelope is generated by linearly expanding or compressing the spectral axis. Each spectral point is identified by its index, ranging from 0 to 511. To transform a spectral profile by 2, the new spectral value at frequency $f$ is generated by averaging the spectral values around the original spectral profile at frequency of $0.5 f$. The transformation process is illustrated in Figure 3. In this figure, an original spectral envelope is being expanded by two. The spectral value at index 150 is thus transformed to spectral index 300 in the new envelope and the original spectral information at high frequencies is lost.

4.   The transformed spectral value is used to resynthesize a speech waveform using the vocal tract excitation information extracted from the original file.

Voice transformation with the STS coder allows listening to transformed speech but requires long computation. We simplified our approach to one of modifying the spectral scale in the spectral domain directly within a mel-scale filterbank analysis program. The incoming speech sample is processed with an FFT to calculate spectral magnitudes. Then spectral magnitudes are linearly transformed. Lastly mel-scale filtering is performed with 10 linearly spaced filters up to 1000 Hz and logarithmically spaced filters from 1000 Hz up. A cosine transform is then used to generate mel-scaled cepstral values that are used by the wordspotter. Much faster processing can be achieved by applying the spectral transformation as part of the filterbank analysis. For example, while performing spectral transformation using the STS algorithm takes up to approximately 10 times real time, spectral transformation within the mel-scale filterbank program can be accomplished within 1/10 real time on a Sparc 10 workstation. The rapid processing rate allows on-line spectral transformation.

## 5   WORD SPOTTING EXPERIMENTS

Linear warping in the spectral domain, which is used in the above algorithm, is correct when the vocal tract is modelled as a series of lossless acoustic tubes and the excitation source is at one end of the vocal tract (Wakita, 1977). Wakita showed that if the vocal tract is modelled as a series of equal length, lossless, and concatenated acoustic tubes, then the ratio of the areas between the tubes determines the relative resonant frequencies of the vocal tract, while the overall length of the vocal tract linearly scales formant frequencies. Preliminary research was conducted using linear scaling with spectral ratios ranging from 0.6 to 1.8 to alter test utterances. After listening to the STS transformed speech and also observing

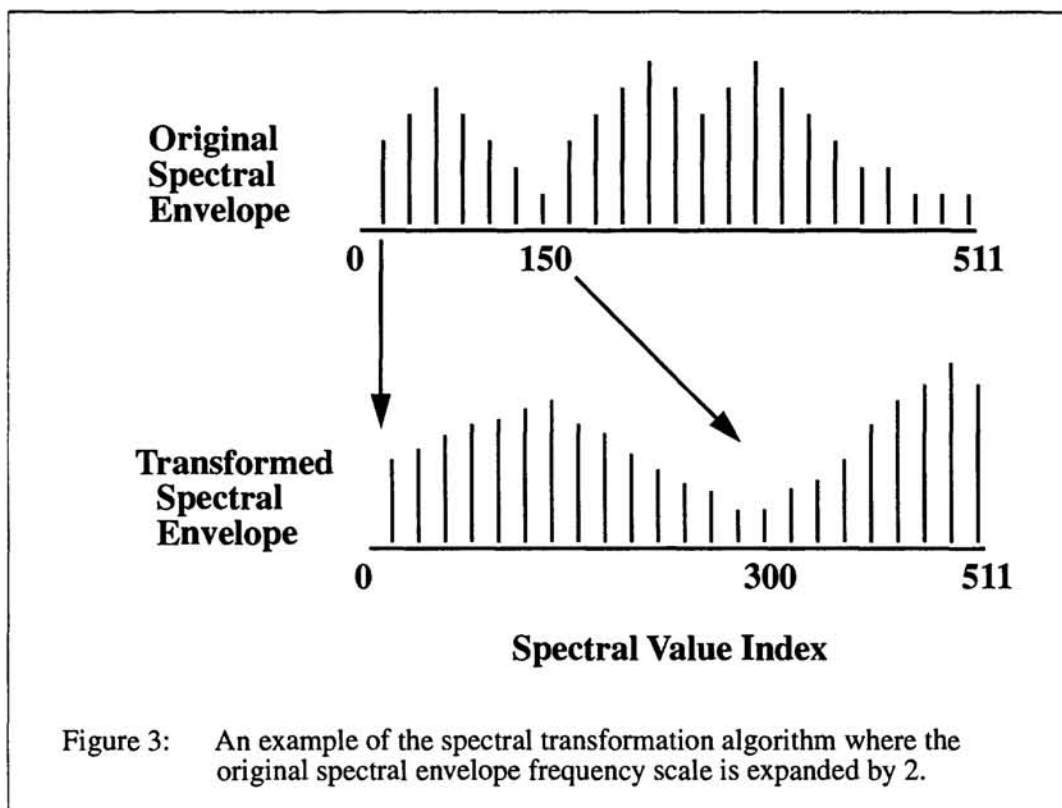

**Spectral Value Index**

Figure 3: An example of the spectral transformation algorithm where the original spectral envelope frequency scale is expanded by 2.

spectrograms of the transformed speech, it was found that speech transformed using ratios between 0.9 and 1.1 are reasonably natural and can represent speech without introducing artifacts.

Using discriminative training techniques such as FOM training carries the risk of overtraining the wordspotter on the training set and obtaining results that are poor on the testing set. To delay the onset of overtraining, we artificially transformed each training set conversation during each epoch using a different random linear transformation ratio.

The transformation ratio used for each conversation is calculated using the following formula: *ratio* $\equiv \alpha + N\,(0, 0.06)$, where $\alpha$ is the transformation ratio that matches each training speaker to the average of the training set speakers, and $N$ is a normally distributed random variable with a mean of 0.0 and standard deviation of 0.06. For each training conversation, the long term averages of formant frequencies for formant 1, 2, and 3 are calculated. A least square estimation is then performed to match the formant frequencies of each training set conversation to the group average formant frequencies. The transformation equation is described below:

$$\begin{bmatrix} \overline{F}_1 \\ \overline{F}_2 \\ \overline{F}_3 \end{bmatrix} = \alpha \bullet \begin{bmatrix} F_1 \\ F_2 \\ F_3 \end{bmatrix}$$

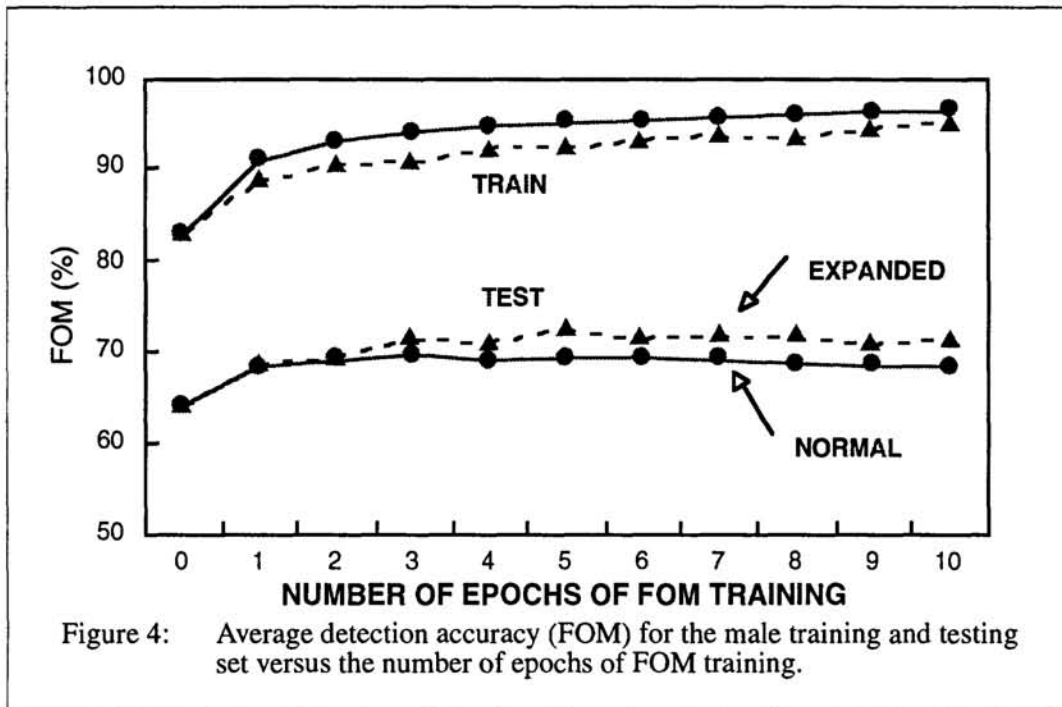

Figure 4:    Average detection accuracy (FOM) for the male training and testing
set versus the number of epochs of FOM training.

The transform ratio for each individual conversation is calculated to improve the natural-
ness of the transformed speech. In preliminary experiments, each conversation was trans-
formed with fixed ratios of 0.9, 0.95, 1.05, and 1.1. However, for a speaker with already
high formant frequencies, pushing the formant frequencies higher may make the trans-
formed speech sound unnatural. By incorporating the individual formant matching ratio
into the transformation ratio, speakers with high formant frequencies are not transformed
to very high frequencies and speakers with low formant frequencies are not transformed to
even lower formant frequency ranges.

Male and female conversations from the NIST credit card database were used separately to
train separate word spotters. Both the male and the female partition of data used 24 conver-
sations for training and 11 conversations for testing. Keyword occurrences were extracted
from each training conversation and used as the data for initialization of the neural network
word spotter. Also, each training conversation was broken up into sentence length segments
to be used for embedded reestimation in which the keyword models are joined with the filler
models and the parameters of all the models are jointly estimated. After embedded reesti-
mation, Figure of Merit training as described in (Lippmann, 1994) was performed for up to
10 epochs. During each epoch, each training conversation is transformed using a transform
ratio randomly generated as described above. The performance of the word spotter after
each iteration of training is evaluated on both the training set and the testing set.

## 6   WORD SPOTTING RESULTS

Training and testing set FOM scores for the male speakers and the female speakers are
shown in Figure 4 and Figure 5 respectively. The $x$ axis plots the number of epochs of FOM
training where each epoch represents presenting all 24 training conversations once. The
FOM for word spotters trained with the normal training conversations and word spotters

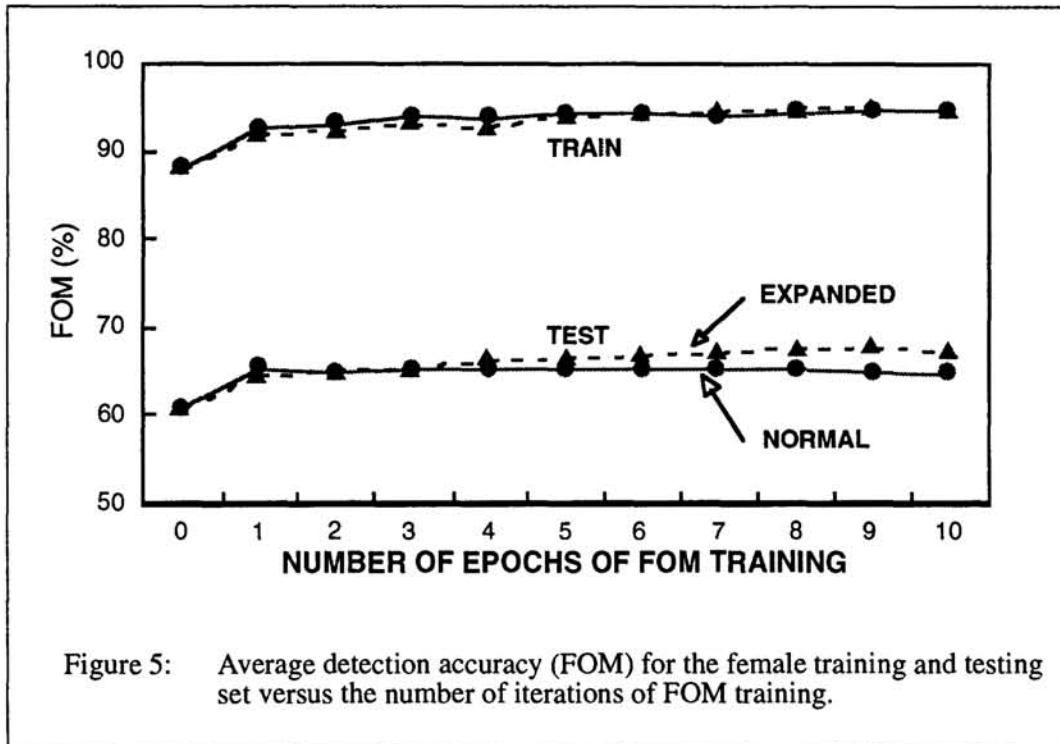

Figure 5: Average detection accuracy (FOM) for the female training and testing set versus the number of iterations of FOM training.

trained with artificially expanded training conversations are shown in each plot. After the first epoch, the FOM improves significantly. With only the original training conversations (normal), the testing set FOM rapidly levels off while the training set FOM keeps on improving.

When the training conversations are artificially expanded, the training set FOM is below the training set FOM from the normal training set due to more difficult training data. However, the testing set FOM continues to improve as more epochs of FOM training are performed. When comparing the FOM of wordspotters trained on the two sets of data after ten epochs of training, the FOM for the expanded set was 2.9 percentage points above the normal FOM for male speakers and 2.5 percentage points above the normal FOM for female speakers. For comparison, Carlson has reported that for a high performance word spotter on this database, doubling the amount of training data typically increases the FOM by 2 to 4 percentage points (Carlson, 1994).

## 7 SUMMARY

Lack of training data has always been a constraint in training speech recognizers. This research presents a voice transformation technique which increases the variety among training talkers. The resulting more varied training set provided up to 2.9 percentage points of improvement in the figure of merit (average detection rate) of a high performance word spotter. This improvement is similar to the increase in performance provided by doubling the amount of training data (Carlson, 1994). This technique can also be applied to other speech recognition systems such as continuous speech recognition, talker identification, and isolated speech recognition.

## ACKNOWLEDGEMENT

This work was sponsored by the Advanced Research Projects Agency. The views expressed are those of the authors and do not reflect the official policy or position of the U.S. Government. We wish to thank Tom Quatieri for providing his sinusoidal transform analysis/synthesis system.

## BIBLIOGRAPHY

B. Carlson and D. Seward. (1994) Diagnostic Evaluation and Analysis of Insufficient and Task-Independent Training Data on Speech Recognition. In *Proceedings Speech Research Symposium XIV,* Johns Hopkins University.

E. Chang and R. Lippmann. (1994) Figure of Merit Training for Detection and Spotting. In *Neural Information Processing Systems 6*, G. Tesauro, J. Cohen, and J. Alspector, (Eds.), Morgan Kaufmann: San Mateo, CA.

H. Drucker, R. Schapire, and P. Simard. (1993) Improving Performance in Neural Networks Using a Boosting Algorithm. In *Neural Information Processing Systems 5*, S. Hanson, J. Cowan, and C. L. Giles, (Eds.), Morgan Kaufmann: San Mateo, California.

N. Iwahashi and Y. Sagisaka. (1994) Speech Spectrum Transformation by Speaker Interpolation. In *Proceedings International Conference on Acoustics Speech and Signal Processing*, Vol. 1, 461-464.

R. Lippmann, E. Chang & C. Jankowski. (1994) Wordspotter Training Using Figure-of-Merit Back Propagation. In *Proceedings of International Conference on Acoustics Speech and Signal Processing*, Vol. 1, 389-392.

H. Mizuno and M. Abe. (1994) Voice Conversion Based on Piecewise Linear Conversion Rules of Formant Frequency and Spectrum Tilt. In *Proceedings International Conference on Acoustics Speech and Signal Processing*, Vol. 1, 469-472.

T. Quatieri and R. McAulay. (1992) Shape Invariant Time-Scale and Pitch Modification of Speech. In *IEEE Trans. Signal Processing*, vol 40, no 3. pp. 497-510.

Hisashi Wakita. (1977) Normalization of Vowels by Vocal-Tract Length and Its Application to Vowel Identification. In *IEEE Trans. Acoustics, Speech, and Signal Processing*, vol. ASSP-25, No. 2., pp. 183-192.